# Probabilistic $n$-Choose-$k$ Models for Classification and Ranking

**Kevin Swersky     Daniel Tarlow**
Dept. of Computer Science
University of Toronto
`[kswersky,dtarlow]@cs.toronto.edu`

**Ryan P. Adams**
School of Eng. and Appl. Sciences
Harvard University
`rpa@seas.harvard.edu`

**Richard S. Zemel**
Dept. of Computer Science
University of Toronto
`zemel@cs.toronto.edu`

**Brendan J. Frey**
Prob. and Stat. Inf. Group
University of Toronto
`frey@psi.toronto.edu`

## Abstract

In categorical data there is often structure in the number of variables that take on each label. For example, the total number of objects in an image and the number of highly relevant documents per query in web search both tend to follow a structured distribution. In this paper, we study a probabilistic model that explicitly includes a prior distribution over such counts, along with a count-conditional likelihood that defines probabilities over all subsets of a given size. When labels are binary and the prior over counts is a Poisson-Binomial distribution, a standard logistic regression model is recovered, but for other count distributions, such priors induce global dependencies and combinatorics that appear to complicate learning and inference. However, we demonstrate that simple, efficient learning procedures can be derived for more general forms of this model. We illustrate the utility of the formulation by exploring applications to multi-object classification, learning to rank, and top-K classification.

## 1   Introduction

When models contain multiple output variables, an important potential source of structure is the number of variables that take on a particular value. For example, if we have binary variables indicating the presence or absence of a particular object class in an image, then the number of "present" objects may be highly structured, such as the number of digits in a zip code. In ordinal regression problems there may be some prior knowledge about the proportion of outputs within each level. For instance, when modeling scores assigned to papers submitted to a conference, this structure can be due to instructions that reviewers assign scores such that the distribution is roughly uniform.

One popular model for multiple output classification problems is logistic regression (LR), in which the class probabilities are modeled as being conditionally independent, given the features; another popular approach utilizes a softmax over the class outputs. Both models can be seen as possessing a prior on the label counts: in the case of the softmax model this prior is explicit that exactly one is active. For LR, there is an implicit factorization in which there is a specific prior on counts; this prior is the source of computational tractability, but also imparts an inductive bias to the model. The starting observation for our work is that we do not lose much efficiency by replacing the LR counts prior with a general prior, which permits the specification of a variety of inductive biases.

In this paper we present a probabilistic model of multiple output classification, the $n$-choose-$k$ model, which incorporates a distribution over the label counts, and show that computations needed

for learning and inference in this model are efficient. We develop applications of this model to diverse problems. A maximum-likelihood version of the model can be used for problems such as multi-class recognition, in which the label counts are known at training time but only a prior distribution is known at test time. The model easily extends to ordinal regression problems, such as ranking or collaborative filtering, in which each item is assigned to one of a small number of relevance levels. We establish a connection between $n$-choose-$k$ models and ranking objectives, and prove that optimal decision theoretic predictions under the model for "monotonic" gain functions (to be defined later), which include standard objectives used in ranking, can be achieved by a simple sorting operation. Other problems can be modeled via direct maximization of expected gain. An important aim in classification and information retrieval is to optimize expected precision@K. We show that we can efficiently optimize this objective under the model and that it yields promising results.

Overall, the result is a class of models along with a well-developed probabilistic framework for learning and inference that makes use of algorithms and modeling components that are not often used in machine learning. We demonstrate that it is a simple, yet expressive probabilistic approach that has many desirable computational properties.

## 2 Binary $n$-Choose-$k$ Model

We begin by defining the basic model under the assumption of binary output variables. In the following section, we will generalize to the case of ordinal variables. The model inputs are $\boldsymbol{x}$, and $\boldsymbol{\theta}$ is defined as $\boldsymbol{\theta} = \mathbf{W}\boldsymbol{x}$, where $\mathbf{W}$ are the parameters. The model output is a vector of $D$ binary variables $\boldsymbol{y} \in \mathcal{Y} = \{0, 1\}^D$. We will use subsets $c \subseteq \{1, \ldots, D\}$ of variable indices and will represent the value assigned to a subset of variables as $\boldsymbol{y}_c$. We will also make use of the notation $\bar{c}$ to mean the complement $\{1, \ldots, D\} \backslash c$. The generative procedure is then defined as follows:

- Draw $k$ from a prior distribution $p(k)$ over counts $k$.
- Draw $k$ variables to take on label 1, where the probability of choosing subset $c$ is given by

$$p(\boldsymbol{y}_c = \mathbf{1}, \boldsymbol{y}_{\bar{c}} = \mathbf{0} \mid k) = \begin{cases} \frac{\exp\{\sum_{d \in c} \theta_d\}}{Z_k(\boldsymbol{\theta})} & \text{if } |c| = k \\ 0 & \text{otherwise} \end{cases}, \qquad (1)$$

where $\boldsymbol{\theta} = (\theta_1, \ldots, \theta_D)$ are parameters that determine individual variable biases towards being off or on, and $Z_k(\boldsymbol{\theta}) = \sum_{\boldsymbol{y} \mid \sum_d y_d = k} \exp\{\sum_d \theta_d y_d\}$. Under this definition $Z_0 = 1$, and $p(\mathbf{0} \mid 0) = 1$. This has been referred to as a conditional Bernoulli distribution [1].

Logistic regression can be viewed as an instantiation of this model, with a "prior" distribution over count values that depends on parameters $\boldsymbol{\theta}$. This is a forced interpretation, but it is useful in understanding the implicit prior over counts that is imposed when using LR. Specifically, if $p(k)$ is defined as be a particular function of $\boldsymbol{\theta}$ (known as a Poisson-Binomial distribution [2]): $p(k; \boldsymbol{\theta}) = \frac{Z_k(\boldsymbol{\theta})}{Z(\boldsymbol{\theta})}$, where $Z(\boldsymbol{\theta}) = \sum_k Z_k(\boldsymbol{\theta})$, then the joint probability $p(\boldsymbol{y}, k; \boldsymbol{\theta})$ becomes equivalent to a LR model in the following sense. Suppose we have a joint assignment of variables $\boldsymbol{y}$ and $\sum_d y_d = k$, and $p(k; \boldsymbol{\theta})$ is Poisson-Binomial, then

$$p(\boldsymbol{y}, k; \boldsymbol{\theta}) = p(k; \boldsymbol{\theta})p(\boldsymbol{y} \mid k; \boldsymbol{\theta}) = \frac{Z_k(\boldsymbol{\theta})}{Z(\boldsymbol{\theta})} \frac{\exp\{\sum_{d \in c} \theta_d\}}{Z_k(\boldsymbol{\theta})} = \prod_d \frac{\exp\{\theta_d y_d\}}{1 + \exp\{\theta_d\}}. \qquad (2)$$

Note that the last equality factorizes $Z(\boldsymbol{\theta})$ to create independence across variables, but it requires that the "prior" be defined in terms of parameters $\boldsymbol{\theta}$. Our interest in this paper is in the more flexible family of models that arise after breaking the dependence of the "prior" on $\boldsymbol{\theta}$. First, we explore treating $p(k)$ as a prior in the Bayesian sense, using it to express prior knowledge about label counts; later we will explore learning $p(k)$ using separate parameters from $\boldsymbol{\theta}$. A consequence of these decisions is that the distribution does not factorize. At this point, we have not made it clear that these models can be learned efficiently, but we will show in the next section that this is indeed the case.

### 2.1 Maximum Likelihood Learning

Our goal in learning is to select parameters so as to maximize the probability assigned to observed data by the model. For notational simplicity in this section, we compute partial derivatives with

respect to $\boldsymbol{\theta}$, then it should be clear that these can be back-propagated to a model of $\boldsymbol{\theta}(\boldsymbol{x}; \mathbf{W})$. We note that if this relationship is linear, and the objective is convex in terms of $\boldsymbol{\theta}$, then it will also be convex in terms of $\mathbf{W}$. The log-likelihood is as follows:

$$\log p(\boldsymbol{y}; \boldsymbol{\theta}) = \log \sum_{k=0}^{D} p(k) p(\boldsymbol{y} \mid k; \boldsymbol{\theta}) = \log p(\boldsymbol{y} \mid \sum_d y_d; \boldsymbol{\theta}) + \kappa \qquad (3)$$

$$= \sum_d \theta_d y_d - \log Z_{\sum_d y_d}(\boldsymbol{\theta}) + \kappa, \qquad (4)$$

where $\kappa$ is a constant that is independent of $\boldsymbol{\theta}$. As is standard, if we are given multiple sets of binary variables, $\{\boldsymbol{y}^n\}_{n=1}^N$, we maximize the sum of log probabilities $\sum_n \log p(\boldsymbol{y}^n; \boldsymbol{\theta})$. The partial derivatives take a standard log-sum-exp form, requiring expectations $\mathbb{E}_{p(y_d \mid k = \sum_{d'} y_{d'})}[y_d]$.

A naive computation of this expectation would require summing over $\binom{D}{k = \sum_d y_d}$ configurations. However, there are more efficient alternatives: the dynamic programming algorithms developed in the context of Poisson-Binomial distributions are applicable, e.g., the algorithm from [3] runs in $O(Dk)$ time. The basic idea is to compute partial sums along a chain that lays out variables $y_d$ in sequence. An alternative formulation of the dynamic program [4] can be made to yield an $O(D \log^2 D)$ algorithm by using a divide-and-conquer algorithm that employs Fast Fourier Transforms (FFTs). These algorithms are quite general and can also be used to compute $Z_k$ values, incorporate prior distributions over count values, and draw a sample of $\boldsymbol{y}$ values conditional upon some $k$ for the same computational cost [5]. We use the FFT tree algorithm from [5] throughout, because it is most flexible and has best worst-case complexity.

## 2.2 Test-time Inference

Having learned a model, we would like to make test-time predictions. In Section 4.2, we will show that optimal decision-theoretic predictions (i.e., that maximize expected gain) can be made in several settings by a simple sorting procedure, and this will be our primary way of using the learned model. However, here, we consider the task of producing a distribution over labels $\boldsymbol{y}$, given $\boldsymbol{\theta}(\boldsymbol{x})$. To draw a joint sample of $\boldsymbol{y}$ values, we can begin by drawing $k$ from $p(k)$, then conditional on that $k$, use the dynamic programming algorithm to draw a sample conditional on $k$.

To compute marginals, a simple strategy is to loop over each value of $k$ and run dynamic programming conditioned on $k$, and then average the results weighted by the respective prior. For priors that only give support to a small number of $k$ values, this is quite efficient. An alternative approach is to draw several samples of $k$ from $p(k)$, then for each sampled value, run dynamic programming to compute marginals. Averaging these marginals can then be seen as a Rao-Blackwellized estimate. Finally, it is possible to compute exact marginals for arbitrary $p(k)$ in a single run of an $O(D \log^2 D)$ dynamic programming algorithm, but the simpler strategies were sufficient for our needs here, so we do not pursue that direction further.

## 3   Ordinal $n$-Choose-$k$ Model

An extension of the binary $n$-choose-$k$ model can be developed in the case of ordinal data, where we assume that labels $\boldsymbol{y}$ can take on one of $R$ categorical labels, and where there is an inherent ordering to labels $R > R - 1 > \ldots > 1$; each label represents a relevance label in a learning-to-rank setting. Let $k_r$ represent the number of variables $\boldsymbol{y}$ that take on label $r$ and define $\boldsymbol{k} = (k_R, \ldots, k_1)$. The idea in the ordinal case is to define a joint model over count variables $\boldsymbol{k}$, then to reduce the conditional distribution of $p(\boldsymbol{y} \mid \boldsymbol{k})$ to be a series of binary models. The generative model is defined as follows:

- Initialize all variables $\boldsymbol{y}$ to be unlabeled.
- Sample $k_R, \ldots, k_1$ jointly from $p(\boldsymbol{k})$.
- Repeat for $r = R$ to 1:
  - Choose a set $c_r$ of $k_r$ unlabeled variables $\boldsymbol{y}_{\leq r}$ and assign them relevance label $r$. Choose subsets with probability equal to the following:

$$p(\boldsymbol{y}_{\leq r, c_r} = \mathbf{1}, \boldsymbol{y}_{\leq r, \bar{c}_r} = \mathbf{0} \mid k_r) = \begin{cases} \frac{\exp\{\sum_{d \in c_r} \theta_d\}}{Z_{r,k}(\boldsymbol{\theta}, \boldsymbol{y}_{\leq r})} & \text{if } |c_r| = k_r \\ 0 & \text{otherwise} \end{cases}, \qquad (5)$$

where we use the notation $\boldsymbol{y}_{\leq r}$ to represent all variables that are given a relevance label less than or equal to $r$. $Z_{r,k}$ is similar to the normalization constant $Z_k$ that appears in the binary model, but it is restricted to sum over $\boldsymbol{y}_{\leq r}$ instead of the full $\boldsymbol{y}$:
$Z_{r,k_r}(\boldsymbol{\theta}, \boldsymbol{y}_{\leq r}) = \sum_{\boldsymbol{y}_{\leq r}|(\sum_d \mathbf{1}\{y_d = r\}) = k_r} \exp\{\theta_d \cdot \mathbf{1}\{y_d = r\}\}$.

Note that if $R = D$ and $p(\boldsymbol{k})$ specifies that $k_r = 1$ for all $r$, then this process defines a Plackett-Luce (PL) [6, 7, 8] ranking model. One interpretation of this model is as a "group" PL model, where instead of drawing individual elements in the generative process, groups of elements are drawn simultaneously. In this work, we focus on ranking with weak labels ($R < D$) which is more restrictive than modeling distributions over permutations [9], where learning would require marginalizing over all possible permutations consistent with the given labels. In this setting, inference in the ordinal $n$-choose-$k$ model is both exact and efficient.

### 3.1 Maximum Likelihood Learning

Let $k_r = \sum_d \mathbf{1}\{y_d = r\}$. The log likelihood of parameters $\boldsymbol{\theta}$ can be written as follows:

$$\log \sum_{\boldsymbol{k} \in \mathcal{K}} p(\boldsymbol{k})p(\boldsymbol{y} \mid \boldsymbol{k}; \boldsymbol{\theta}) = \sum_{r=1}^{R} \left[ \sum_{d:y_d=r} \theta_d - \log Z_{r,k_r}(\boldsymbol{\theta}, \boldsymbol{y}_{\leq r}) \right] + \kappa. \tag{6}$$

Here, we see that learning decomposes into the sum of $R$ objectives that are of the same form as arise in the binary $n$-choose-$k$ model. As before, the only non-trivial part of the gradient computation comes from the log-sum-exp term, but the required expectations that arise can be efficiently computed using dynamic programming. In this case, $R - 1$ calls are required.

### 3.2 Test-time Inference

The test-time inference procedure in the ordinal model is similar to the binary case. Brute force enumeration over $\boldsymbol{k}$ becomes exponentially more expensive as $R$ grows, but for some priors where $p(\boldsymbol{k})$ has sparse support, this may be feasible. To draw samples of $\boldsymbol{y}$, the main requirement is the ability to draw a joint sample of $\boldsymbol{k}$ from $p(\boldsymbol{k})$. In the case that $p(\boldsymbol{k})$ is a simple distribution such as a multinomial, this can be done easily. It is also possible to efficiently draw a joint sample if the distribution over $\boldsymbol{k}$ takes the form $p(\boldsymbol{k}) = \mathbf{1}\{\sum_r k_r = D\} \cdot \prod_r p(k_r)$. That is, there is an arbitrary but independent prior over each $k_r$ value, along with a single constraint that the chosen $k_r$ values sum to exactly $D$. Given a sample of $\boldsymbol{k}$, it is straightforward to sample $\boldsymbol{y}$ using $R$ calls to dynamic programming. To do so, begin by using the binary algorithm to sample $k_R$ variables to take on value $R$. Then remove the chosen variables from the set of possible variables, and sample $k_{R-1}$ variables to take on value $R - 1$. Repeat until all variables have been assigned a value.

An alternative to producing marginal probabilities at test time is trying to optimize performance under a task-specific evaluation measure. The main motivation for the ordinal model is the learning to rank problem [10], so our main interest is in methods that do well under such task-specific evaluation measures that arise in the ranking task. In Section 4.2, we show that we can make exact optimal decision theoretic test-time predictions under the learning-to-rank gain functions without the need for sampling.

## 4 Incorporating Gain

### 4.1 Training to Maximize Expected Top-K Classification Gain

One of the motivating applications for this model is the top-K classification (TKC) task. We formulate this task using a gain function, parameterized by a value $K$ and a "scoring vector" $\boldsymbol{t}$, which is assumed to be of the same dimension as $\boldsymbol{y}$. The gain function stipulates that $K$ elements of $\boldsymbol{y}$ are chosen, (assigning a score of zero if some other number is chosen), and assigns reward for choosing each element of $\boldsymbol{y}$ based on $\boldsymbol{t}$. Specifically the gain function is defined as follows:

$$G_K(\boldsymbol{y}, \boldsymbol{t}) = \begin{cases} \sum_d y_d t_d & \text{if } \sum_d y_d = K \\ 0 & \text{otherwise}. \end{cases} \tag{7}$$

The same gain can be used for Precision@K, in which case the number of nonzero values in $\boldsymbol{t}$ is unrestricted. Here, we focus on the case where $\boldsymbol{t}$ is binary with a single nonzero entry at index $d^*$.

An interesting issue is what gain function should be used to train a model when the test-time evaluation metric is TKC, or Precision@K. Maximum likelihood training of TKC in this case of a single target class could correspond to a version of our $n$-choose-$k$ model in which $p(k)$ is a spike at $k = 1$; note that in this case the $n$-choose-$k$ model is equivalent to a softmax over the output classes. An alternative is to train using the same gain function used at test-time.

Here, we consider incorporating the TKC gain at training time for binary $\boldsymbol{t}$ with one nonzero entry, training the model to maximize expected gain. Specifically, the objective is the following:

$$\mathbb{E}_p[G_K(\boldsymbol{y}, \boldsymbol{t})] = \sum_k \sum_{\boldsymbol{y}} p(k)p(\boldsymbol{y} \mid k)\mathbf{1}\left\{\sum_d y_d = K\right\}\sum_d y_d t_d = \sum_{\boldsymbol{y}} p(K)p(\boldsymbol{y} \mid K)y_{d^*} \quad (8)$$

It becomes clear that this objective is equivalent to the marginal probability of $y_{d^*}$ under a prior distribution that places all its mass on $k = K$. In Section 5.3, we empirically investigate training under expected gain versus training under maximum likelihood

## 4.2 Optimal Decision-theoretic Predictions for Monotonic Gain Functions

We now turn attention to gain functions defined on *rankings* of items. Letting $\pi$ be a permutation, we define a "monotonic" gain function as follows:

**Definition 1.** *A gain function $G(\pi, \boldsymbol{r})$ is a* monotonic ranking gain *if:*

- *It can be expressed as $\sum_{d=1}^{D} \alpha_d f(r_{\pi_d})$, where $\alpha_d$ is a weighting (or discount) term, and $\pi_d$ is the index of the item ranked in position d,*

- *$\alpha_d \geq \alpha_{d+1} \geq 0$ for all d, and*

- *$f(r) \geq f(r-1) \geq 0$ for all $r \geq r'$.*

It is straightforward to see that popular learning-to-rank scoring functions like normalized discounted cumulative gain (NDCG) and Precision@K are monotonic ranking gains. $NDCG(\pi, \boldsymbol{r}) \propto \sum_d \frac{2^{r_{\pi_d}}-1}{\log_2(1+d)}$, so set $\alpha_d = \kappa \cdot \frac{1}{\log_2(1+d)}$ and $f(r) = 2^r - 1$. We define Precision@K gain to be the fraction of documents in the top K produced ranks that have label $R$: $P@K(\pi, \boldsymbol{r}) = \sum_d \mathbf{1}\{d \leq K\}\mathbf{1}\{r_{\pi_d} = R\}$, so set $\alpha_d = \mathbf{1}\{d \leq K\}$ and $f(r) = \mathbf{1}\{r = R\}$.

The expected gain under a monotonic ranking gain and ordinal $n$-choose-$k$ model is

$$\mathbb{E}_p[G(\pi)] = \sum_{\boldsymbol{y}' \in \mathcal{Y}} p(\boldsymbol{y}')\sum_{d=1}^{D}\alpha_d f(y'_{\pi_d}) = \sum_{d=1}^{D}\alpha_d \sum_{y'_{\pi_d}=1}^{R} f(y'_{\pi_d})p(y_{\pi_d} = y'_{\pi_d}) = \sum_{d=1}^{D}\alpha_d g_{\pi_d}, \quad (9)$$

where we have defined $g_d = \sum_{r=1}^{R} f(r)p(y_d = r)$.

We now state four propositions and a lemma. The proofs of the propositions mostly result from algebraic manipulation, so we leave their proof to the supplementary materials. The main theorem will be proved afterwards.

**Proposition 1.** *If $\theta_i \geq \theta_j$, then $p(y_i = R) \geq p(y_j = R)$.*

**Proposition 2.** *If $\theta_i \geq \theta_j$ and $p(y_i \geq r) \geq p(y_j \geq r)$, then $p(y_i \geq r - 1) \geq p(y_j \geq r - 1)$.*

**Lemma 1.** *If $\theta_i \geq \theta_j$, then for all $r$, $p(y_i \geq r) \geq p(y_j \geq r)$.*

*Proof.* By induction. Proposition 1 is the base case, and Proposition 2 is the inductive step. □

**Proposition 3.** *If $\theta_i \geq \theta_j$ and $f$ is defined as in Definition 1, then $g_i \geq g_j$.*

**Proposition 4.** *Consider two pairs of non-negative real numbers $a_i$, $a_j$ and $b_i$, $b_j$ where $a_i \geq a_j$ and $b_i \geq b_j$. It follows that $a_i b_i + a_j b_j \geq a_i b_j + a_j b_i$.*

**Theorem 1.** *Under an ordinal $n$-choose-$k$ model, the optimal decision theoretic predictions for a monotonic ranking gain are made by sorting $\boldsymbol{\theta}$ values.*

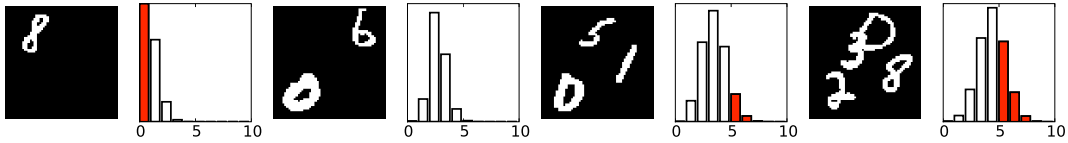

Figure 1: Four example images from the embedded MNIST dataset test set, along with the Poisson-Binomial distribution produced by logistic regression for each image. The area marked in red has zero probability under the data distribution, but the logistic regression model is not flexible enough to model it.

*Proof.* Without loss of generality, assume that we are given a vector $\boldsymbol{\alpha}$ corresponding to placing the $\alpha$'s in descending order and a vector $\boldsymbol{g}_\pi$ where $\pi$ is some arbitrary ordering of the $g$'s. The goal now is to find the ordering $\pi^*$ that maximizes the objective given in (9) which is equivalently expressed as the inner product $\boldsymbol{\alpha}^T \boldsymbol{g}_\pi$.

Assume that we are given an ordering $\hat{\pi}$ where for at least one pair $i$, $j$ where $i > j$, we have that $\theta_{\hat{\pi}_i} < \theta_{\hat{\pi}_j}$. Furthermore, assume that this ordering is optimal. That is, $\hat{\pi} = \pi^*$. By Proposition 3 we have that $g_{\hat{\pi}_i} < g_{\hat{\pi}_j}$. The contributions of these elements to the overall objective is given by $\alpha_i g_{\hat{\pi}_i} + \alpha_j g_{\hat{\pi}_j}$. By Proposition 4 we improve the objective by swapping $\theta_{\hat{\pi}_i}$ and $\theta_{\hat{\pi}_j}$ contradicting the assumption that $\hat{\pi}$ is a local optimum.

If we have multiple elements that are not in sorted order, then we can repeat this argument by considering pairs of elements until the whole vector is sorted. □

## 5 Experiments

### 5.1 Modeling Varying Numbers of Objects

Our first experiment explores an issue that arises frequently in computer vision, where there are an unknown number of objects in an image, but the number is highly structured. We developed a multiple image dataset that simulates this scenario.[1] To generate an image, we uniformly sampled a count between 1 and 4, and then take that number of digit instances (with at most one instance per digit class) from the MNIST dataset and embed them in a $60 \times 60$ image. The $x, y$ locations are chosen from a $4 \times 4$ uniformly spaced grid and and then a small amount of jitter is added. We generated 10,000 images each for the training, test, and validation sets. The goal is to predict the set of digits that appear in a given image. Examples can be seen in Figure 1.

We train a binary $n$-choose-$k$ model on this dataset. The inputs to the model are features learned from the images by a standard Restricted Boltzmann Machine with 1000 hidden units. As a baseline, we trained a logistic regression classifier on the features and achieved a test-set negative log-likelihood (NLL) of $2.84$. Ideally, this model should learn that there are never more than four digits in any image. In Figure 1, we show four test images, and the Poisson-Binomial distribution over counts that arises from the logistic regression model. Marked in red are regions where there is zero probability of the count value in the data distribution. Here it is clear that the implicit count prior in LR is not powerful enough to model this data. As a comparison, we trained a binary $n$-choose-$k$ model where we explicitly parameterize and learn an input-dependent prior. The model learns the correct distribution over counts and achieves a test-set NLL of $1.95$. We show a visualization of the learned likelihood and prior parameters in the supplementary material.

### 5.2 Ranking

A second set of experiments considers learning-to-rank applications of the $n$-choose-$k$ model. We report on comparisons to other ranking approaches, using seven datasets associated with the LETOR 3.0 benchmark [10]. Following the standard LETOR procedures, we trained over five folds, each with distinct training, validation, and testing splits.

For each dataset, we train an ordinal $n$-choose-$k$ model to maximize the likelihood of the data, where each training example consists of a number of items, each assigned a particular relevance level; the number of levels ranges from 2-4 across the datasets. At test time, we produce a *ranking*, which as

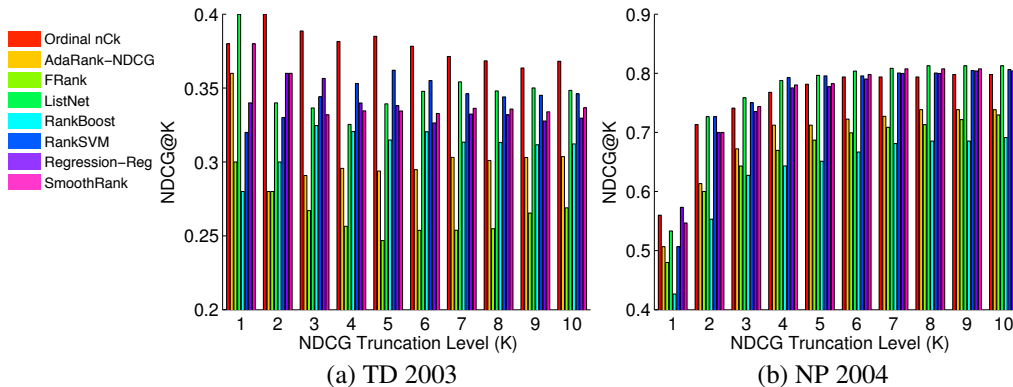

Figure 2: Ranking results on two datasets from LETOR 3.0. Results for the other 5 datasets, along with Precision@K results, appear in the supplementary material.

shown in Section 4.2 is the optimal decision theoretic prediction under a ranking gain function, by simply sorting the items for each test query based on their $\theta$ score values. Note that this is a very simple ranking model, in that the score assigned to each test item by the model is a linear function of the input features, and the only hyperparameter to tune is an $\ell_2$ regularization strength.

Results for two of the data sets are shown in Figure 2 (first is our best relative performance, second is typical); the full set of results are in the supplementary material. Several publicly available baselines are shown for comparison. As can be seen in the graphs, our approach is competitive with the state-of-the-art on all data sets, and substantially outperforms all baselines on the TD 2003 dataset. Note that the performance of the baseline methods is quite variable and it appears that overfitting is an issue on these datasets, even for linear models. We hypothesize that proper probabilistic incorporation of weak labels helps to mitigate this effect to some degree.

### 5.3 Top-K Classification

Our third and final set of experiments concern top-K classification, an important task that has gained considerable attention recently in the ImageNet Challenge.[2] Here we consider a task analogous to that in the ImageNet Challenge, in which each image contains a single object label, but a model is allowed to return up to K class predictions per image. A classification is deemed correct if the appropriate class is one of the K returned classes.

We train binary $n$-choose-$k$ models, experimenting with different training protocols that directly maximize expected gain under the model, as described in Section 4.1. That is, we train on the expected top-K gain for different values of $K$. Note that top-1 is equivalent to softmax regression. For each model/evaluation criterion combination, we find the $\ell_2$ penalty that gives the highest validation accuracy; the corresponding test-set results are shown in Table 1. For comparison, we also include logistic regression, where each output is conditionally independent. We experimented on the embedded MNIST dataset where all but one label from each example was randomly removed, and on the Caltech-101 Silhouettes dataset [11], which consists of images of binarized silhouettes from 101 different categories. In both datasets we trained the models using the pixels as inputs. We noticed that the optimal $\ell_2$ strength chosen by each method was quite high, suggesting that overfitting is an issue in these datasets. When the $\ell_2$ strength is low, the difference between the objectives becomes more apparent. On Caltech it is clear that training for the expected gain improves the corresponding test accuracy in this regime. On the embedded MNIST dataset, when the $\ell_2$ strength is low there is a surprising result that the top-3 and top-5 criteria outperform top-1, even when top-1 is used as the evaluation measure. Since there are several digits actually present in the ground truth, there is no real signal in the data that differentiates the digit labeled as the target from the other equally valid "distractor" digits. In order to satisfy the top-1 objective for the given target, the learning algorithm is forced to find some arbitrary criterion by which to cause the given target to be preferred over the distractors, which is harmful for generalization purposes. This scenario does occur in datasets like ImageNet, where multiple objects can be present in a single image. It would be interesting to repeat these experiments on more challenging, large scale datasets, but we leave this for future work.

|  | Top 1 / Top 3 / Top 5 | Top 1 / Top 3 / Top 5 | Top 1 / Top 3 / Top 5 | Top 1 / Top 3 / Top 5 |
|---|---|---|---|---|
| LR | 0.606 / 0.785 / 0.812 | 0.545 / 0.716 / 0.766 | 0.346 / 0.647 / 0.815 | 0.263 / 0.557 / 0.742 |
| Top 1 | 0.621 / 0.796 / 0.831 | 0.574 / 0.755 / 0.804 | 0.353 / 0.659 / 0.820 | 0.268 / 0.569 / 0.757 |
| Top 3 | 0.614 / 0.792 / 0.834 | 0.558 / 0.771 / 0.813 | 0.353 / 0.671 / 0.834 | 0.318 / 0.637 / 0.815 |
| Top 5 | 0.602 / 0.787 / 0.834 | 0.523 / 0.767 / 0.823 | 0.330 / 0.659 / 0.824 | 0.313 / 0.642 / 0.822 |
| | (a) Caltech Sil. strong $\ell_2$ | (b) Caltech Sil. weak $\ell_2$ | (c) EMNIST strong $\ell_2$ | (d) EMNIST weak $\ell_2$ |

Table 1: Top-K classification results when various models are trained using an expected top-K gain and then tested using some possibly different top-K criterion. The rows correspond to training criteria, and the columns correspond to test criteria. (a) and (c) show the test accuracy when a strong $\ell_2$ regularizer is used, while (b) and (d) use a relatively weaker regularizer. Logistic regression is included for comparison.

# 6 Related Work

Our work here is related to many different areas; we cannot hope to survey all related work in multi-label classification and ranking. Instead, we focus on work related to the main novelty in this paper, the explicit modeling of structure on label counts. That is, given that we have prior knowledge of label count structure, or are modeling a domain that exhibits such structure, the question is how can the structure be leveraged to improve a model.

The first and most direct approach is the one that we take here: explicitly model the count structure *within* the model. There are other alternative approaches that are similar in this respect. The work of [12] considers MAP inference in the context of cardinality-based models and develops applications to named entity recognition tasks. Similarly, [13] develops an example application where a cardinality-based term constrains the number of pixels that take on the label "foreground" in a foreground/background image segmentation task. [14] develops models that include a penalty in the energy function for using more labels, which can be seen as a restricted form of structure over label cardinalities.

An alternative way of incorporating structure over counts into a model is via the gain function. The work of Joachims [15] can be seen in this light – the training objective is formulated so as to optimize performance on evaluation measures that include Precision@K. A different approach to including count information in the gain function comes from [16], which trains an image segmentation model so as match count statistics present in the ground truth data. Finally, there are other approaches that do not neatly fall into either category, such as the posterior regularization framework of [17] and related works such as [18]. There, structure, including structure that encodes prior knowledge about counts, such as there being at least one verb in most sentences, is added as a regularization term that is used both during learning and during inference.

Overall, the main difference between our work and these others is that we work in a proper probabilistic framework, either maximizing likelihood, maximizing expected gain, and/or making proper decision-theoretic predictions at test time. Importantly, there is no significant penalty for assuming the proper probabilistic approach: learning is exact, and test-time prediction is efficient.

# 7 Discussion

We have presented a flexible probabilistic model for multiple output variables that explicitly models structure in the number of variables taking on specific values. The model is simple, efficient, easy to learn due to its convex objective, and widely applicable. Our theoretical contribution provides a link between this type of ordinal model and ranking problems, bridging the gap between the two tasks, and allowing the same model to be effective for several quite different problems. Finally, there are many extensions. More powerful models of $\boldsymbol{\theta}$ can be put into the formulation, and gradients can easily be back-propagated. Also, while we chose to take a maximum likelihood approach in this paper, the model is well suited to fully Bayesian inference using e.g., slice sampling. The unimodal posterior distribution should lead to good behavior of the sampler. Beyond these extensions, we believe the framework here to be a valuable modeling building block that has broad application to problems in machine learning.

## Footnotes

[1]http://www.cs.toronto.edu/~kswersky/data/

[2]http://www.image-net.org/challenges/LSVRC/2011/

## References

[1] S. X. Chen and J. S. Liu. Statistical applications of the Poisson-Binomial and conditional Bernoulli distributions. *Statistica Sinica*, 7(4), 1997.

[2] X. H. Chen, A. P. Dempster, and J. S. Liu. Weighted finite population sampling to maximize entropy. *Biometrika*, 81(3):457–469, 1994.

[3] M. H. Gail, J. H. Lubin, and L. V. Rubinstein. Likelihood calculations for matched case-control studies and survival studies with tied death times. *Biometrika*, 68:703–707, 1981.

[4] L. Belfore. An O(n) log2(n) algorithm for computing the reliability of k-out-of-n:G and k-to-l-out-of-n:G systems. *IEEE Transactions on Reliability*, 44(1), 1995.

[5] D. Tarlow, K. Swersky, R. Zemel, R.P. Adams, and B. Frey. Fast exact inference for recursive cardinality models. In *Uncertainty in Artificial Intelligence*, 2012.

[6] R. Plackett. The analysis of permutations. *Applied Statistics*, pages 193–202, 1975.

[7] R.D. Luce. *Individual Choice Behavior a Theoretical Analysis*. Wiley, 1959.

[8] J. Guiver and E. Snelson. Bayesian inference for plackett-luce ranking models. In *International Conference on Machine Learning*, 2009.

[9] J. Huang, C. Guestrin, and L. Guibas. Efficient inference for distributions on permutations. In *Advances in Neural Information Processing Systems*, 2007.

[10] T. Qin, T.Y. Liu, J. Xu, and H. Li. LETOR: A benchmark collection for research on learning to rank for information retrieval. *Information Retrieval Journal*, 2010.

[11] B. Marlin, K. Swersky, B. Chen, and N. de Freitas. Inductive principles for restricted Boltzmann machine learning. In *Artificial Intelligence and Statistics*, 2010.

[12] R. Gupta, A. Diwan, and S. Sarawagi. Efficient inference with cardinality-based clique potentials. In *International Conference on Machine Learning*, 2007.

[13] D. Tarlow, I. Givoni, and R. Zemel. HOP-MAP: Efficient message passing for high order potentials. In *Artificial Intelligence and Statistics*, 2010.

[14] A. Delong, A. Osokin, H.N. Isack, and Y. Boykov. Fast approximate energy minimization with label costs. *International Journal of Computer Vision*, 96(1):127, 2012.

[15] T. Joachims. A support vector method for multivariate performance measures. In *International Conference on Machine Learning*, 2005.

[16] P. Pletscher and P. Kohli. Learning low-order models for enforcing high-order statistics. In *Artificial Intelligence and Statistics*, 2012.

[17] K. Ganchev, J. Graça, J. Gillenwater, and B. Taskar. Posterior regularization for structured latent variable models. *Journal of Machine Learning Research*, 11:2001–2049, 2010.

[18] G. Mann and A McCallum. Generalized expectation criteria with application to semi-supervised classification and sequence modeling. *Journal of Machine Learning Research*, 11:955–984, 2010.

